# Reasoning about Time and Knowledge in Neural-Symbolic Learning Systems

**Artur S. d'Avila Garcez**$^\delta$ **and Luis C. Lamb**$^\lambda$
$^\delta$Dept. of Computing, City University London
London, EC1V 0HB, UK (aag@soi.city.ac.uk)
$^\lambda$Dept. of Computing Theory, PPGC-II-UFRGS
Porto Alegre, RS 91501-970, Brazil (lamb@inf.ufrgs.br)

## Abstract

We show that temporal logic and combinations of temporal logics and modal logics of knowledge can be effectively represented in artificial neural networks. We present a *Translation Algorithm* from temporal rules to neural networks, and show that the networks compute a fixed-point semantics of the rules. We also apply the translation to the muddy children puzzle, which has been used as a testbed for distributed multi-agent systems. We provide a complete solution to the puzzle with the use of simple neural networks, capable of reasoning about time and of knowledge acquisition through inductive learning.

## 1 Introduction

Hybrid neural-symbolic systems concern the use of problem-specific symbolic knowledge within the neurocomputing paradigm (d'Avila Garcez et al., 2002a). Typically, translation algorithms from a symbolic to a connectionist representation and vice-versa are employed to provide either (*i*) a neural implementation of a logic, (*ii*) a logical characterisation of a neural system, or (*iii*) a hybrid learning system that brings together features from connectionism and symbolic artificial intelligence (Holldobler, 1993).

Until recently, neural-symbolic systems were not able to fully represent, reason and learn expressive languages other than propositional and fragments of first-order logic (Cloete & Zurada, 2000). However, in (d'Avila Garcez et al., 2002b; d'Avila Garcez et al., 2002c; d'Avila Garcez et al., 2003), a new approach to knowledge representation and reasoning in neural-symbolic systems based on neural networks ensembles has been introduced. This new approach shows that *modal logics* can be effectively represented in artificial neural networks.

In this paper, following the approach introduced in (d'Avila Garcez et al., 2002b; d'Avila Garcez et al., 2002c; d'Avila Garcez et al., 2003), we move one step further and show that temporal logics can be effectively represented in artificial neural

$^0$Artur Garcez is partly supported by the Nuffield Foundation. Luis Lamb is partly supported by CNPq. The authors would like to thank the referees for their comments.

networks. This is done by providing a translation algorithm from temporal logic theories to the initial architecture of a neural network. A theorem then shows that the translation is correct by proving that the network computes a fixed-point semantics of its corresponding temporal theory (van Emden & Kowalski, 1976). The result is a new learning system capable of reasoning about knowledge and time. We have validated the *Connectionist Temporal Logic* (*CTL*) proposed here by applying it to a distributed time and knowledge representation problem known as the *muddy children puzzle* (Fagin et al., 1995).

*CTL* provides a combined (multi-modal) connectionist system of knowledge and time, which allows the modelling of evolving situations such as changing environments or possible worlds. Although a number of multi-modal systems - e.g., combining knowledge and time (Halpern & Vardi, 1986; Halpern et al., 2003) and combining beliefs, desires and intentions (Rao & Georgeff, 1998) - have been proposed for distributed knowledge representation, little attention has been paid to the integration of a learning component for knowledge acquisition. This work contributes to bridge this gap by allowing the knowledge representation to be integrated in a neural learning system. Purely from the point of view of knowledge representation in neural-symbolic systems, this work contributes to the long term aim of representing expressive and computationally well-behaved symbolic formalisms in neural networks.

The remainder of this paper is organised as follows. We start, in Section 2, by describing the muddy children puzzle, and use it to exemplify the main features of *CTL*. In Section 3, we formally introduce *CTL*'s *Translation Algorithm*, which maps knowledge and time theories into artificial neural networks, and prove that the translation is correct. In Section 4, we conclude and discuss directions for future work.

## 2  Connectionist Reasoning about Time and Knowledge

Temporal logic and its combination with other modalities such as knowledge and belief operators have been the subject of intense investigation (Fagin et al., 1995). In this section, we use the muddy children puzzle, a testbed for distributed knowledge representation formalisms, to exemplify how knowledge and time can be expressed in a connectionist setting. We start by stating the puzzle (Fagin et al., 1995; Huth & Ryan, 2000).

There is a number $n$ of (truthful and intelligent) children playing in a garden. A certain number of children $k$ ($k \leq n$) has mud on their faces. Each child can see if the other are muddy, but not themselves. Now, consider the following situation: A caretaker announces that at least one child is muddy ($k \geq 1$) and asks *does any of you know if you have mud on your own face*? To help understanding the puzzle, let us consider the cases in which $k = 1$, $k = 2$ and $k = 3$. If $k = 1$ (only one child is muddy), the muddy child answers *yes* at the first instance since she cannot see any other muddy child. All the other children answer *no* at the first instance. If $k = 2$, suppose children 1 and 2 are muddy. At the first instance, all children can only answer *no*. This allows 1 to reason as follows: if 2 had said *yes* the first time, she would have been the only muddy child. Since 2 said *no*, she must be seeing someone else muddy; and since I cannot see anyone else muddy apart from 2, I myself must be muddy! Child 2 can reason analogously, and also answers *yes* the second time round. If $k = 3$, suppose children 1, 2 and 3 are muddy. Every children can only answer *no* the first two times round. Again, this allows 1 to reason as follows: if 2 or 3 had said *yes* the second time, they would have been the only two muddy children. Thus, there must be a third person with mud. Since I can only see 2 and

3 with mud, this third person must be me! Children 2 and 3 can reason analogously to conclude as well that *yes*, they are muddy.

The above cases clearly illustrate the need to distinguish between an agent's *individual knowledge* and *common knowledge* about the world in a particular situation. For example, when $k = 2$, after everybody says *no* at the first round, it becomes common knowledge that at least two children are muddy. Similarly, when $k = 3$, after everybody says *no* twice, it becomes common knowledge that at least three children are muddy, and so on. In other words, when it is common knowledge that there are at least $k-1$ muddy children; after the announcement that nobody knows if they are muddy or not, then it becomes common knowledge that there are at least $k$ muddy children, for if there were $k-1$ muddy children all of them would know that they had mud in their faces.[1]

In what follows, a modality $\mathbf{K}_j$ is used to represent the knowledge of an agent $j$. In addition, the term $p_i$ is used to denote that proposition $p$ is *true* for agent $i$. For example, $\mathbf{K}_j \mathrm{p}_i$ means that agent $j$ knows that $p$ is *true* for agent $i$. We use $p_i$ to say that child $i$ is muddy, and $q_k$ to say that at least $k$ children are muddy ($k \leq n$). Let us consider the case in which three children are playing in the garden ($n = 3$). Rule $r_1^1$ below states that when child 1 knows that at least one child is muddy and that neither child 2 nor child 3 are muddy then child 1 knows that she herself is muddy. Similarly, rule $r_2^1$ states that if child 1 knows that there are at least two muddy children and she knows that child 2 is not muddy then she must also be able to know that she herself is muddy, and so on. The rules for children 2 and 3 are interpreted analogously.

| | |
|---|---|
| $r_1^1$: $\mathbf{K}_1\mathrm{q}_1 \wedge \mathbf{K}_1\neg\mathrm{p}_2 \wedge \mathbf{K}_1\neg\mathrm{p}_3 \rightarrow \mathbf{K}_1\mathrm{p}_1$ | $r_2^1$: $\mathbf{K}_1\mathrm{q}_2 \wedge \mathbf{K}_1\neg\mathrm{p}_2 \rightarrow \mathbf{K}_1\mathrm{p}_1$ |
| $r_3^1$: $\mathbf{K}_1\mathrm{q}_2 \wedge \mathbf{K}_1\neg\mathrm{p}_3 \rightarrow \mathbf{K}_1\mathrm{p}_1$ | $r_4^1$: $\mathbf{K}_1\mathrm{q}_3 \rightarrow \mathbf{K}_1\mathrm{p}_1$ |

Table 1: Snapshot rules for agent(child) 1

Each set of snapshot rules $r_m^l$ ($1 \leq l \leq n$; $m \in \mathbb{N}^+$) can be implemented in a single hidden layer neural network $\mathcal{N}_l$ as follows. For each rule, a hidden neuron is created. Each rule antecedent (e.g., $\mathbf{K}_1\mathrm{q}_1$ in $r_1^1$) is associated with an input neuron. The rule consequent ($\mathbf{K}_1\mathrm{p}_1$) is associated with an output neuron. Finally, the input neurons are connected to the output neuron through the hidden neuron associated with the rule ($r_1^1$). In addition, weights and biases need to be set up to implement the meaning of the rule. When a neuron is activated (i.e. has activation above a given threshold), we say that its associated concept (e.g., $\mathbf{K}_1\mathrm{q}_1$) is *true*. Conversely, when a neuron is not activated, we say that its associated concept is *false*. As a result, each input vector of $\mathcal{N}_l$ can be associated with an *interpretation* (an assignment of truth-values) to the set of rules. Weights and biases must be such that the output neuron is activated if and only if the interpretation associated with the input vector satisfies the rule antecedent. In the case of rule $r_1^1$, the output neuron associated with $\mathbf{K}_1\mathrm{p}_1$ must be activated (*true*) if the input neuron associated with $\mathbf{K}_1\mathrm{q}_1$, the input neuron associated with $\mathbf{K}_1\neg\mathrm{p}_2$, and the input neuron associated with $\mathbf{K}_1\neg\mathrm{p}_3$ are all activated (*true*).

The *Connectionist Inductive Learning and Logic Programming (C-ILP) System* (d'Avila Garcez et al., 2002a; d'Avila Garcez & Zaverucha, 1999) makes use of the above kind of translation. *C-ILP* is a massively parallel computational model based on an artificial neural network that integrates inductive learning from examples and background knowledge with deductive learning through logic programming. Follow-

ing (Holldobler & Kalinke, 1994) (see also (Holldobler et al., 1999)), a *Translation Algorithm* maps any logic program $\mathcal{P}$ into a single hidden layer neural network $\mathcal{N}$ such that $\mathcal{N}$ computes the least fixed point of $\mathcal{P}$. This provides a massively parallel model for computing the stable model semantics of $\mathcal{P}$ (Lloyd, 1987). In addition, $\mathcal{N}$ can be trained with examples using, e.g., *Backpropagation*, and using $\mathcal{P}$ as background knowledge (Pazzani & Kibler, 1992). The knowledge acquired by training can then be extracted (d'Avila Garcez et al., 2001), closing the learning cycle (as in (Towell & Shavlik, 1994)).

For each agent (child), a *C-ILP* network can be created. Each network can be seen as representing a (learnable) possible world containing information about the knowledge held by an agent in a distributed system. Figure 1 shows the implementation of rules $r_1^1$ to $r_4^1$. In addition, it contains output neurons $p_1{}^2$ and $\mathbf{K}q_1$, $\mathbf{K}q_2$ and $\mathbf{K}q_3$, all represented as *facts*.[3] This is highlighted in grey in Figure 1. Neurons that appear on both the input and output layers of a *C-ILP* network (e.g., $\mathbf{K}q_1$) are recurrently connected using weight one, as depicted in Figure 1. This allows the network to iterate the computation of truth-values when chains occur in the set of rules. For example, if $a \to b$ and $b \to c$ are rules of the theory, neuron $b$ will appear on both the input and output layers of the network, and if $a$ is activated then $c$ will be activated through the activation of $b$.

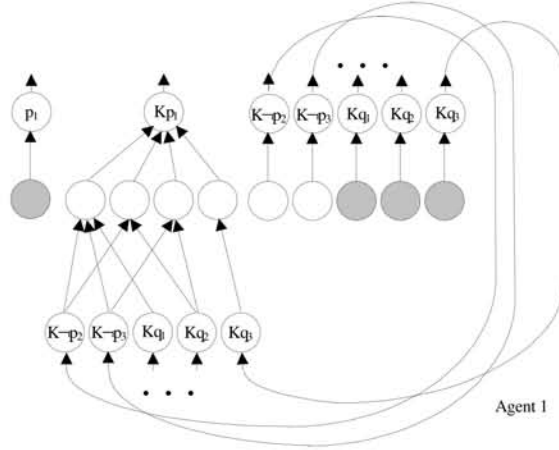

Figure 1: The implementation of rules $\{r_1^1, ..., r_4^1\}$.

If child 1 is muddy, output neuron $p_1$ must be activated. Since, child 2 and 3 can see child 1, they will know that $p_1$ is muddy. This can be represented as $p_1 \to K_2 p_1$ and $p_1 \to K_3 p_1$, and analogously for $p_2$ and $p_3$. This means that the activation of output neurons $K_1 \neg p_2$ and $K_1 \neg p_3$ in Figure 1 depends on the activation of neurons that are not in this network ($\mathcal{N}_1$), but in $\mathcal{N}_2$ and $\mathcal{N}_3$. We need, therefore, to model how the networks in the ensemble interact with each other.

Figure 2 illustrates the interaction between three *C-ILP* networks in the muddy children puzzle. The arrows connecting the networks implement the fact that when a child is muddy, the other children can see her. So if, e.g., neuron $p_1$ is activated in $\mathcal{N}_1$, neuron $\mathbf{K}p_1$ must be activated in $\mathcal{N}_2$ and $\mathcal{N}_3$. For the sake of clarity, the snapshot rules $r_m^1$ shown in Figure 1 are omitted here, and this is indicated in Figure

2 by neurons highlighted in black. In addition, only positive information about the problem is shown in Figure 2. Negative information such as $\neg p_1$, $\mathbf{K}\neg p_1$, $\mathbf{K}\neg p_2$ and $\mathbf{K}\neg p_3$ would be implemented analogously.

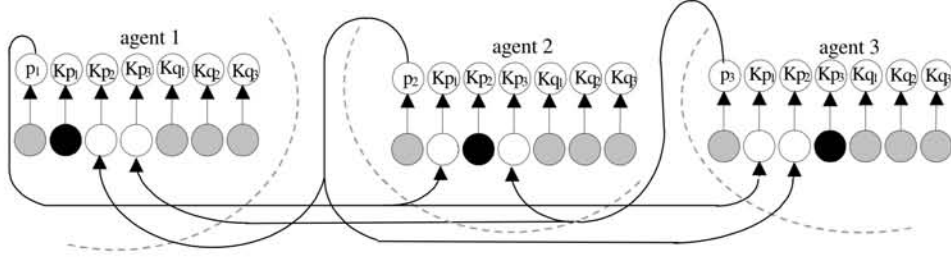

Figure 2: Interaction between agents in the muddy children puzzle.

Figure 2 illustrates well the idea behind this paper. By combining a number of simple *C-ILP* networks, we are able to model individual and common knowledge. Each network represents a possible world or an agent's current set of beliefs (d'Avila Garcez et al., 2002b). If we allow a number of ensembles like the one of Figure 2 to be combined, we can represent the evolution in time of an agent's set of beliefs. This is exactly what is required for a complete solution of the muddy children puzzle, as discussed below.

As we have seen, the solution to the muddy children puzzle illustrated in Figures 1 and 2 considers only *snapshots* of knowledge evolution along time rounds without the addition of a time variable (Huth & Ryan, 2000). A complete solution, however, requires the addition of a temporal variable to allow reasoning about the knowledge acquired after each time round. The snapshot solution of Figures 1 and 2 should then be seen as representing the knowledge held by the agents at an arbitrary time $t$. The knowledge held by the agents at time $t + 1$ would then be represented by another set of *C-ILP* networks, appropriately connected to the original set of networks. Let us consider again the case where $k = 3$. There are alternative ways of representing that, but one possible representation for child 1 would be as follows:

$$t_1 : \neg K_1 p_1 \wedge \neg K_2 p_2 \wedge \neg K_3 p_3 \rightarrow \bigcirc K_1 q_2$$
$$t_2 : \neg K_1 p_1 \wedge \neg K_2 p_2 \wedge \neg K_3 p_3 \rightarrow \bigcirc K_1 q_3$$

Table 2: Temporal rules for agent(child) 1

Each temporal rule is labelled by a time point $t_i$ in which the rule holds. In addition, if a rule labelled $t_i$ makes use of the *next time* temporal operator $\bigcirc$ then whatever $\bigcirc$ qualifies refers to the next time $t_{i+1}$ in a linear time flow. As a result, the first temporal rule above states that if, at $t_1$, no child knows whether she is muddy or not then, at $t_2$, child 1 will know that at least two children are muddy. Similarly, the second rule states that, at $t_2$, if still no child knows whether she is muddy or not then, at $t_3$, child 1 will know that at least three children are muddy. As before, analogous temporal rules exist for agents (children) 2 and 3. The temporal rules, together with the snapshot rules, provide a complete solution to the puzzle. This is depicted in Figure 3 and discussed below.[4]

In Figure 3, networks are replicated to represent an agent's knowledge evolution in time. A network represents an agent's knowledge today (or at $t_1$), a network repre-

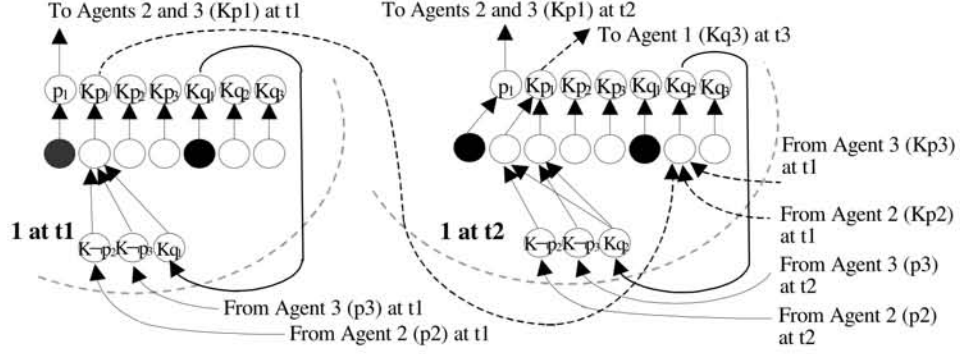

Figure 3: Knowledge evolution of agent (child) 1 from time $t_1$ to time $t_2$.

sents the same agent's knowledge tomorrow ($t_2$), and the appropriate connections between networks model the relations between today and tomorrow according to $\bigcirc$. In the case of $t_1 : \neg K_1 p_1 \wedge \neg K_2 p_2 \wedge \neg K_3 p_3 \rightarrow \bigcirc K_1 q_2$, for example, output neuron $K_1 p_1$ of the network that represents agent 1 at $t_1$, output neuron $K_2 p_2$ of the network that represents agent 2 at $t_1$, and output neuron $K_3 p_3$ of the network that represents agent 3 at $t_1$ need to be connected to output neuron $K_1 q_2$ of the network that represents agent 1 at $t_2$ (the next time) such that $K_1 q_2$ is activated if $K_1 p_1$, $K_2 p_2$ and $K_3 p_3$ are not activated. In conclusion, in order to represent time, in addition to knowledge, we need to use a two-dimensional $C\text{-}ILP$ ensemble. In one dimension we encode the knowledge interaction between agents at a given time point, and in the other dimension we encode the agents' knowledge evolution through time.

## 3  Temporal Translation Algorithm

In this section, we present an algorithm to translate temporal rules of the form $t : \bigcirc K_a L_1, ..., \bigcirc K_b L_k \rightarrow \bigcirc K_c L_{k+1}$, where $a, b, c...$ are agents and $1 \leq t \leq n$,[5] into (two-dimensional) $C\text{-}ILP$ network ensembles. Let $\mathcal{P}$ represent a number $q$ of ground[6] temporal rules. In such rules, we call $L_i$ ($1 \leq i \leq k+1$) a *literal*, and call $K_j L_i$ ($1 \leq j \leq m$) an *annotated literal*. Each $L_i$ can be either a positive literal ($p$) or a negative literal ($\neg p$). Similarly, $K_j L_i$ can be preceded by $\neg$. We use $A_{min}$ to denote the minimum activation for a neuron to be considered active (*true*), $A_{min} \in (0, 1)$. We number the (annotated) literals[7] of $\mathcal{P}$ from 1 to $v$ such that, when a $C\text{-}ILP$ network $\mathcal{N}$ is *created*, the input and output layers of $\mathcal{N}$ are vectors of length $v$, where the i-th neuron represents the i-th (annotated) literal. For convenience, we use a bipolar semi-linear activation function $h(x) = \frac{2}{1 + e^{-\beta x}} - 1$, and inputs in $\{-1, 1\}$.

Let $k_l$ denote the number of (annotated) literals in the body of rule $r_l$; $\mu_l$, the number of rules in $\mathcal{P}$ with the same (annotated) literal as consequent, for each rule $r_l$; $MAX_{r_l}(k_l, \mu_l)$, the greater element between $k_l$ and $\mu_l$ for rule $r_l$; and $MAX_{\mathcal{P}}(k_1, ..., k_q, \mu_1, ..., \mu_q)$, the greatest element among all $k_l$'s and $\mu_l$'s of $\mathcal{P}$. We

also use $\overrightarrow{k}$ as a shorthand for $(k_1, ..., k_q)$, and $\overrightarrow{\mu}$ as a shorthand for $(\mu_1, ..., \mu_q)$. For example, for $\mathcal{P} = \{r_1 : b \wedge c \wedge \neg d \rightarrow a, r_2 : e \wedge f \rightarrow a, r_3 : \rightarrow b\}$, $k_1 = 3$, $k_2 = 2$, $k_3 = 0$, $\mu_1 = 2$, $\mu_2 = 2$, $\mu_3 = 1$, $MAX_{r_1}(k_1, \mu_1) = 3$, $MAX_{r_2}(k_2, \mu_2) = 2$, $MAX_{r_3}(k_3, \mu_3) = 1$ and $MAX_{\mathcal{P}}(\overrightarrow{k}, \overrightarrow{\mu}) = 3$.

**CTL Translation Algorithm:**

1. For each time point $t$ in $\mathcal{P}$ do: For each agent $j$ in $\mathcal{P}$ do: Create a *C-ILP* Neural Network $\mathcal{N}_{j,t}$.

2. Calculate $W$ such that $W \geq \frac{2}{\beta} \cdot \frac{\ln(1+A_{\min}) - \ln(1-A_{\min})}{MAX_{\mathcal{P}}(\overrightarrow{k}, \overrightarrow{\mu}) \cdot (A_{\min}-1) + A_{\min}+1}$;

3. For each rule in $\mathcal{P}$ of the form $t : \bigcirc K_1 L_1, ..., \bigcirc K_{m-1} L_k \rightarrow \bigcirc K_m L_{k+1},$[8] do: (a) Add a hidden neuron $L^{\bigcirc}$ to $\mathcal{N}_{m,t+1}$ and set $h(x)$ as the activation function of $L^{\bigcirc}$; (b) Connect each neuron $\bigcirc K_j L_i$ $(1 \leq i \leq k)$ in $\mathcal{N}_{j,t}$ to $L^{\bigcirc}$. If $L_i$ is a positive (annotated) literal then set the connection weight to $W$; otherwise, set the connection weight to $-W$. Set the threshold $\theta_l^{\bigcirc}$ of $L^{\bigcirc}$ to $\theta_l^{\bigcirc} = \frac{(1+A_{\min})(k_l-1)}{2} W$; (c) Connect $L^{\bigcirc}$ to $K_m L_{k+1}$ in $\mathcal{N}_{m,t+1}$ and set the connection weight to $W$. Set the threshold $\theta_l^{t+1}$ of $K_m L_{k+1}$ to $\theta_l^{t+1} = \frac{(1+A_{\min})(1-\mu_l)}{2} W$; (d) Add a hidden neuron $L^{\bullet}$ to $\mathcal{N}_{m,t}$ and set $h(x)$ as the activation function of $L^{\bullet}$; (e) Connect neuron $K_m L_{k+1}$ in $\mathcal{N}_{m,t+1}$ to $L^{\bullet}$ and set the connection weight to $W$; Set the threshold $\theta_l^{\bullet}$ of $L^{\bullet}$ to *zero*; (f) Connect $L^{\bullet}$ to $\bigcirc K_m L_{k+1}$ in $\mathcal{N}_{m,t}$ and set the connection weight to $W$. Set the threshold $\theta_l^t$ of $K_m L_{k+1}$ to $\theta_l^t = \frac{(1+A_{\min})(1-\mu_l)}{2} W$;

4. For each rule in $\mathcal{P}$ of the form $t : \bigcirc K_1 L_1, ..., \bigcirc K_{m-1} L_k \rightarrow K_m L_{k+1}$, do: (a) Add a hidden neuron $L^{\bigcirc}$ to $\mathcal{N}_{m,t}$ and set $h(x)$ as the activation function of $L^{\bigcirc}$; (b) Connect each neuron $\bigcirc K_j L_i$ $(1 \leq i \leq k)$ in $\mathcal{N}_{j,t}$ to $L^{\bigcirc}$. If $L_i$ is a positive (annotated) literal then set the connection weight to $W$; otherwise, set the connection weight to $-W$. Set the threshold $\theta_l^{\bigcirc}$ of $L^{\bigcirc}$ to $\theta_l^{\bigcirc} = \frac{(1+A_{\min})(k_l-1)}{2} W$; (c) Connect $L^{\bigcirc}$ to $K_m L_{k+1}$ in $\mathcal{N}_{m,t}$ and set the connection weight to $W$. Set the threshold $\theta_l^{t+1}$ of $K_m L_{k+1}$ to $\theta_l^{t+1} = \frac{(1+A_{\min})(1-\mu_l)}{2} W$;

5. If $\mathcal{N}$ ought to be fully-connected, set all other connections to zero.

In the above algorithm it is worth noting that, whenever a rule consequent is preceded by $\bigcirc$, a forward connection from $t$ to $t+1$ and a feedback connection from $t+1$ to $t$ need to be added to the ensemble. For example, if $t : a \rightarrow \bigcirc b$ is a rule of $\mathcal{P}$ then not only must the activation of neuron $a$ at $t$ activate neuron $b$ at $t+1$, but the activation of neuron $b$ at $t+1$ must also activate neuron $\bigcirc b$ at $t$. This is implemented in steps $3(d)$ to $3(f)$ of the algorithm. The remainder of the algorithm is concerned with the implementation of snapshot rules (as in Figure 1). The values of $W$ and $\theta$ come from *C-ILP's* Translation Algorithm (d'Avila Garcez & Zaverucha, 1999), and are chosen so that the behaviour of the network matches that of the temporal rules, as the following theorem shows.

**Theorem 1** *(Correctness of Translation Algorithm) For each set of ground temporal rules $\mathcal{P}$, there exists a neural network ensemble $\mathcal{N}$ such that $\mathcal{N}$ computes the fixed-point operator $T_{\mathcal{P}}$ of $\mathcal{P}$.*

**Proof.** *(sketch) This proof follows directly from the proof of the analogous theorem for single* C-ILP *networks presented in (d'Avila Garcez & Zaverucha, 1999). This is so because* C-ILP*'s definition for $W$ and $\theta$ values makes hidden neurons $L^{\bigcirc}$ and $L^{\bullet}$ behave like* and *gates, while output neurons behave like* or *gates.* $\square$

# 4    Conclusions

In his seminal paper (Valiant, 1984), Valiant argues for the need of rich logic-based knowledge representation mechanisms within learning systems. In this paper, we have addressed such a need, yet complying with important principles of connectionism such as massive parallelism. In particular, a very important feature of the system presented here (CTL) is the temporal dimension that can be combined with an epistemic dimension. This paper provides the first account of how to integrate such dimensions in a neural-symbolic learning system. The CTL framework opens up several interesting research avenues in the domain of neural-symbolic integration, allowing for the representation and learning of expressive formalisms. In this paper, we have illustrated this by providing a full solution to the muddy children puzzle, where agents reason about their knowledge at different time points. In the near future, we plan to also apply the system to a large, real world case study.

## Footnotes

[1]Notice that this reasoning process can only start once it is common knowledge that at least one child is muddy, as announced by the caretaker.

[2]Note $p_1$ means 'child 1 is muddy' while $\mathbf{K}p_1$ means 'child 1 knows she is muddy'.

[3]A fact is normally represented as a rule with no antecedents. *C-ILP* represents facts by not connecting the rule's hidden neuron to any input neuron (in the case of fully-connected networks, weights with initial value zero are used).

[4]It is worth noting that each network remains a simple, single hidden layer neural network that can be trained with the use of standard *Backpropagation* or other off-the-shelf learning algorithm.

[5]There may be $n + 1$ time points since, e.g., $t_1 : K_j \alpha, K_k \beta \rightarrow \bigcirc K_j \gamma$ means that if agent $j$ knows $\alpha$ and agent $k$ knows $\beta$ at time $t_1$ then agent $j$ knows $\gamma$ at time $t_2$.

[6]Variables such as $t_i$ are instantiated into the language's ground terms ($t_1, t_2, t_3...$).

[7]We use '*(annotated) literals*' to refer to any literal, annotated or not annotated ones.

[8] Note that $\bigcirc$ is not required to precede every rule antecedent. In the network, neurons are labelled as $\bigcirc K_1 L_1$ or $K_1 L_1$ to differentiate the two concepts.

# References

Cloete, I., & Zurada, J. M. (Eds.). (2000). *Knowledge-based neurocomputing*. The MIT Press.

d'Avila Garcez, A. S., Broda, K., & Gabbay, D. M. (2001). Symbolic knowledge extraction from trained neural networks: A sound approach. *Artificial Intelligence, 125*, 155–207.

d'Avila Garcez, A. S., Broda, K., & Gabbay, D. M. (2002a). *Neural-symbolic learning systems: Foundations and applications*. Perspectives in Neural Computing. Springer-Verlag.

d'Avila Garcez, A. S., Lamb, L. C., Broda, K., & Gabbay, D. M. (2003). Distributed knowledge representation in neural-symbolic learning systems: a case study. *Accepted for Proceedings of 16th International FLAIRS Conference*. St. Augustine Florida.

d'Avila Garcez, A. S., Lamb, L. C., & Gabbay, D. M. (2002b). *A connectionist inductive learning system for modal logic programming* (Technical Report 2002/6). Department of Computing, Imperial College, London.

d'Avila Garcez, A. S., Lamb, L. C., & Gabbay, D. M. (2002c). A connectionist inductive learning system for modal logic programming. *Proceedings of IEEE International Conference on Neural Information Processing ICONIP'02* (pp. 1992–1997). Singapore.

d'Avila Garcez, A. S., & Zaverucha, G. (1999). The connectionist inductive learning and logic programming system. *Applied Intelligence Journal, Special Issue on Neural Networks and Structured Knowledge, 11*, 59–77.

Fagin, R., Halpern, J., Moses, Y., & Vardi, M. (1995). *Reasoning about knowledge*. MIT Press.

Halpern, J. Y., van der Meyden, R., & Vardi, M. Y. (2003). Complete axiomatizations for reasoning about knowledge and time. *SIAM Journal on Computing*. to appear.

Halpern, J. Y., & Vardi, M. (1986). The complexity of reasoning about knowledge and time I: lower bounds. *Journal of Computer and System Sciences, 38*, 195–237.

Holldobler, S. (1993). Automated inferencing and connectionist models. Postdoctoral Thesis, Intellektik, Informatik, TH Darmstadt.

Holldobler, S., & Kalinke, Y. (1994). Toward a new massively parallel computational model for logic programming. *Proceedings of the Workshop on Combining Symbolic and Connectionist Processing, ECAI 94* (pp. 68–77).

Holldobler, S., Kalinke, Y., & Storr, H. P. (1999). Approximating the semantics of logic programs by recurrent neural networks. *Applied Intelligence Journal, Special Issue on Neural Networks and Structured Knowledge, 11*, 45–58.

Huth, M. R. A., & Ryan, M. D. (2000). *Logic in computer science: Modelling and reasoning about systems*. Cambridge University Press.

Lloyd, J. W. (1987). *Foundations of logic programming*. Springer-Verlag.

Pazzani, M., & Kibler, D. (1992). The utility of knowledge in inductive learning. *Machine Learning, 9*, 57–94.

Rao, A. S., & Georgeff, M. P. (1998). Decision procedures for BDI logics. *Journal of Logic and Computation, 8*, 293–343.

Towell, G. G., & Shavlik, J. W. (1994). Knowledge-based artificial neural networks. *Artificial Intelligence, 70*, 119–165.

Valiant, L. G. (1984). A theory of the learnable. *Communications of the ACM, 27*, 1134–1142.

van Emden, M. H., & Kowalski, R. A. (1976). The semantics of predicate logic as a programming language. *Journal of the ACM, 23*, 733–742.
